# MMCows: A Multimodal Dataset for Dairy Cattle Monitoring

**Hien Vu**
Purdue University
hienvu@purdue.edu

**Omkar Prabhune**
Purdue University
oprabhun@purdue.edu

**Unmesh Raskar**
University of Wisconsin–Madison
uraskar@wisc.edu

**Dimuth Panditharatne**
University of Wisconsin–Madison
panditharatn@wisc.edu

**Hanwook Chung**
Iowa State University
hwchung@iastate.edu

**Christopher Y. Choi**
University of Wisconsin–Madison
cchoi22@wisc.edu

**Younghyun Kim**
Purdue University
younghyun@purdue.edu

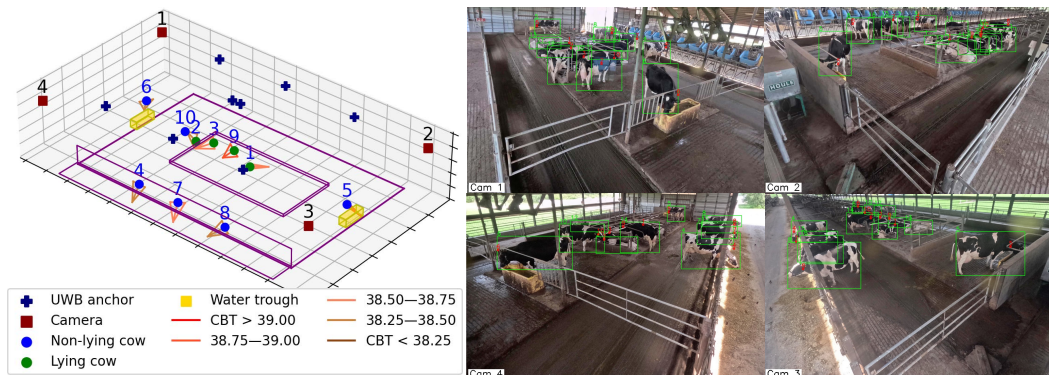

Figure 1: Sample 3D visualization of the MMCows multimodal sensing and four camera views.

## Abstract

Precision livestock farming (PLF) has been transformed by machine learning (ML), enabling more precise and timely interventions that enhance overall farm productivity, animal welfare, and environmental sustainability. However, despite the availability of various sensing technologies, few datasets leverage multiple modalities, which are crucial for developing more accurate and efficient monitoring devices and ML models. To address this gap, we present MMCows, a multimodal dataset for dairy cattle monitoring. This dataset comprises a large amount of synchronized, high-quality measurement data on behavioral, physiological, and environmental factors. It includes two weeks of data collected using wearable and implantable sensors deployed on ten milking Holstein cows, such as ultra-wideband (UWB) sensors, inertial sensors, and body temperature sensors. In addition, it features 4.8 million frames of high-resolution image sequences from four isometric view cameras, as well as temperature and humidity data from environmental sensors. We also gathered milk yield data and outdoor weather conditions. One full day's worth of image data is annotated as ground truth, totaling 20,000 frames with 213,000 bounding boxes of 16 cows, along with their 3D locations and

behavior labels. An extensive analysis of MMCows is provided to evaluate the modalities individually and their complementary benefits. The release of MMCows and its benchmarks will facilitate research on multimodal monitoring of dairy cattle, thereby promoting sustainable dairy farming. The dataset and the code for benchmarks are available at `https://github.com/neis-lab/mmcows`.

# 1   Introduction

The dairy industry around the world is under strong sustainability pressure—environmentally, socially, and economically. Environmentally, massive consumption of clean water and energy and the waste produced by cattle threaten the environmental sustainability of the industry [1]. At the same time, strong social pressure is placed on farmers to raise cattle in a more humane way [2]. However, due to the low economic margin of the dairy industry, dairy farms tend to house more cows in large-scale facilities often with inadequate living conditions [3], making it even more challenging to carry out environment- and animal-friendly farming. These compounded problems also result in significant production losses which have been estimated at several billions of US dollars annually, along with excessive costs in energy and water usage, that increasingly and negatively affect dairy producers worldwide [4, 5, 6]. In short, the sustainability of the dairy industry depends on the maintenance of larger herds at minimal labor costs with low water and energy consumption, while keeping them healthy and stress-free.

Precision agriculture, or precision livestock farming (PLF) more specifically, has emerged as an effective solution to address these sustainability challenges in dairy farming [4]. Powered by advances in sensing, computing, and communication technologies [7], PLF enables the monitoring of individual animals' behavior, physiology, and their social interaction in real time to quickly detect health problems and better track their diet, growth, and productivity [8, 5], allowing fine-grained monitoring and control of facilities [9]. More recently, similar to many other application domains, PLF has seen a remarkable transformation with the advent of machine learning (ML) techniques [10, 11]. Combined with various sensing and computer vision techniques, several ML approaches have been proposed to monitor livestock animals. Examples include computer vision-based measurement of cow body weight [12] and detection of bovine respiratory diseases using wearable inertial measurement units (IMUs) [13].

With the rapid rise in the popularity of ML approaches and enabled by the increasing availability of low-cost high-quality sensors, a number of datasets of dairy cattle have been introduced. Each modality has different pros and cons in terms of accuracy, cost, animal friendliness, etc., and to determine the optimal modality or complementary modalities for a target application, a high-quality multimodal dataset is crucial. Unfortunately, existing dairy cattle datasets with only one or two sensing modalities do not meet the needs of recent ML research.

In this work, we present the MMCows dataset that leverages the complementary benefits of synchronized data from multiple modalities for accurate and efficient monitoring of dairy cattle. The dataset was collected from Holstein cows, which is the dominant breed of dairy cattle worldwide [14]. MMCows also includes data related to the environment and milk yield, which can be used to provide a comprehensive understanding of behavior and physiological changes of the cattle over time. The contributions and unique aspects of the MMCows dataset are as follows:

- **Multiple sensing modalities:** MMCows is a large-scale fine-grained dataset of dairy cattle collected over a two-week period featuring multiple sensing modalities. The variety of modalities ranges from physiological and behavioral sensing to visual, environmental, and milk yield data. As ground truth, MMCows contains one day's worth of annotated visual data of 16 cows, including 20,000 isometric-view images with IDs and behavior labels.

- **Real-world data:** The data was collected from actual milking cows housed in the Agricultural Research Station at the University of Wisconsin–Madison. The deployment was carefully planned and executed without disrupting the cows' daily routine or negatively affecting their comfort to obtain physiological and behavioral data that is as natural as possible, similar to what can be observed at commercial dairy farms.

- **Comprehensive data and extensive benchmarks:** MMCows not only contains primary measurement data, but also various secondary processed data derived from it. We also present

a comprehensive set of benchmarks that utilize both primary and secondary data for various applications of cattle monitoring.

## 2 Related Work

In this section, we discuss advanced sensing technologies that enable the collection of various data required for PLF, and the lack of suitable multimodal datasets required for ML research.

### 2.1 Health monitoring of dairy cattle

Behavioral and physiological responses in dairy cattle are widely utilized in both research and practice for detecting health issues, and the monitoring of such responses is critical for the timely detection of various health conditions including heat stress, lameness, ketosis, mastitis, estrus, and calving.

Heat stress occurs when a cow's core body temperature exceeds the upper critical threshold of the thermal neutral zone, leading to behavioral changes [15]. Cows experiencing heat stress tend to stand more to increase surface area for better cooling through convection [16, 17, 18]. They also reduce the number of meals per day to lower metabolic heat production [19, 20, 21, 22], and their milk production decreases as a result [23, 24, 25]. Additionally, cows under heat stress drink more frequently but in smaller amounts [26].

Lameness in dairy cattle is characterized by abnormal gait or movement due to pain or injury in the limbs or feet that significantly impacts well-being and productivity [27]. Lame cows typically exhibit reduced daily feeding time and fewer feeding visits, coupled with an increased feeding rate [28, 29]. Accurate prediction of lameness was achieved by combining data on neck acceleration and milk production [30].

Ketosis is a metabolic disorder where energy demands exceed intake, that results in a negative energy balance, leading to rapid reductions in daily milk yield, feeding time, and feeding rate in affected cows [28, 31].

Mastitis, an inflammatory condition of the udder, is one of the most economically significant diseases in dairy cattle. Cows with mastitis show decreased feeding and ruminating time, alongside increased idle standing time [32].

Estrus and calving can be detected early through behavioral monitoring. Indicators of estrus include standing heat, intense physical activity, and mounting behaviors [33, 34]. Before calving, cows typically increase their daily step count and reduce lying and feeding time within the 24 hours leading up to calving [35, 36].

### 2.2 Behavioral and physiological sensing of dairy cattle

Behavioral and physiological responses of dairy cattle have been widely used to detect health issues in research and practice. For behavior monitoring, the most common methods involve measuring cows' movements and locations. An accelerometer mounted on the neck or ankle provides useful information about a cow's body movements and postures [37, 38, 39, 40]. While this approach allows for accurate behavior inference, wearable devices can be relatively costly to deploy and maintain at scale. Cow location data also offers insights into their activities (such as feeding and drinking) and social interactions, typically using UWB or GPS for localization [41, 42, 43]. UWB provides precise locations but requires an infrastructure of stationary anchor devices, whereas GPS functions without such infrastructure but has relatively low accuracy, especially indoors. High-power consumption and the need for wearable sensors are drawbacks of both localization methods.

An increasingly popular solution for tracking cattle movement and location is ML-based vision processing. Various vision models have been proposed to identify individual cows [44, 45, 46] and to recognize behavior and posture [47]. Vision models can also be applied for localization [48]. The primary advantage of vision-based approaches is that they eliminate the need for wearable devices, which simplifies deployment and improves animal comfort. However, these methods are susceptible to changes in lighting conditions and physical obstructions.

One of the most critical physiological data is core body temperature (CBT) due to its relevance to animal behavior and health conditions, particularly heat stress [17, 49, 8, 50]. CBT can be measured

Table 1: Comparison of related datasets.

| | Datasets | # of modal. | Main modalities UWB | IMU | RGB | # of cams. | # of frames | Duration | # of subj. | Annotation classes ID | Behavior |
|---|---|---|---|---|---|---|---|---|---|---|---|
| Human | Stanford-ECM [58] | 3 | - | ✓ | ✓ | 1 | - | 31h | 10 | - | 24 |
| | ActionSense [59] | 8 | - | ✓ | ✓ | 7 | 512k | 13h | 10 | - | 20 |
| | mRI [60] | 4 | - | ✓ | ✓ | 1 | 160k | 0.3h | 20 | - | 12 |
| | SALSA [61] | 5 | - | - | ✓ | 4 | - | 1h | 18 | - | - |
| Cattle and swine | PBVD-5 [62] | 1 | - | - | ✓ | 1 | - | 8d | 9 | 9 | 5 |
| | Zhang et al. [63] | 1 | - | - | ✓ | 1 | - | 1.5h | 12 | - | 5 |
| | Bergamini et al. [64] | 2 | - | - | ✓ | 1 | 3.4M | 23d | 8 | 8 | 5 |
| | FriesianCattle2015 [65] | 1 | - | - | ✓ | 1 | 764 | - | 92 | - | - |
| | FriesianCattle2017 [48] | 1 | - | - | ✓ | 1 | 940 | 2h | 89 | - | - |
| | AerialCattle2017 [48] | 1 | - | - | ✓ | 1 | 16k | 0.2h | 23 | 23 | - |
| | Ter-Sarkisov et al. [66] | 1 | - | - | ✓ | - | - | 14d | 10 | - | - |
| | DSCOW [42] | 1 | ✓ | - | - | - | - | 123d | 190 | - | - |
| | Rodriguez et al. [67] | 1 | - | ✓ | - | - | - | 28d | 20 | - | 7 |
| | OpenCows2020 [14] | 1 | - | - | ✓ | 1 | 3.7k | - | 46 | - | - |
| | Cows2021 [68] | 1 | - | - | ✓ | 1 | 10k | - | 186 | - | - |
| | Koskela et al. [69] | 1 | - | - | ✓ | 1 | 1.7M | 19h | - | - | 7 |
| | CowScreeningDB [70] | 1 | - | ✓ | - | - | - | 7h | 43 | - | - |
| | **MMCows** (ours) | 9 | ✓ | ✓ | ✓ | 4 | 4.8M | 14d | 16 | 16 | 7 |

using commercial temperature sensors inserted in the vagina or rectum, which is considered the conventional "gold standard" method [51, 52]. However, the insertion and retrieval process is very costly and stressful for both farmers and cows, and due to the depth of implantation, real-time measurement is not feasible. Ingestible boluses have been used to measure the reticulum temperature [53], but their measurement results can be affected by water and feed intake, making them unsuitable for precise CBT measurement. More recent studies have shown that subcutaneously injected temperature sensors can be used for real-time CBT measurement [54, 55, 56, 57], though they are not yet commercially available.

## 2.3 Related datasets

As seen in Section 2.2, each sensing modality has different pros and cons in terms of accuracy, cost, animal friendliness, etc. To develop cattle monitoring devices and ML models that are accurate, cost-effective, and animal-friendly, a careful evaluation of different modalities or combinations of modalities must be performed, which requires an appropriate dataset. Table 1 compares various datasets developed for animal (cattle or swine) and human subjects.

For the monitoring of human subjects, which have been relatively well studied, several datasets are available for multimodal ML, including [58, 59, 60]. In these datasets, RGB images are the most common, and IMUs are also widely used to record joint movements. Other sensing modalities found in human datasets include heart rate, audio, depth, mmWave, etc. [58, 60, 61].

On the other hand, although many animal datasets are available for ML research, none of them were developed mainly for multimodal ML. As shown in Table 1, most datasets consist of RGB images, usually close-up top-view images that are useful for identification of the animals [65, 48, 66, 14, 68, 69, 62, 63, 64]. Other datasets contain IMU data for motion detection and behavior classification [67, 70] or UWB data for localization [42], but not both. Only one dataset [64] includes an additional modality, depth images, alongside traditional RGB images; however, both fall into the category of image sensing. As a result, these datasets are only suitable for developing and evaluating ML models for a certain modality and do not provide a way to compare different modalities and their combinations.

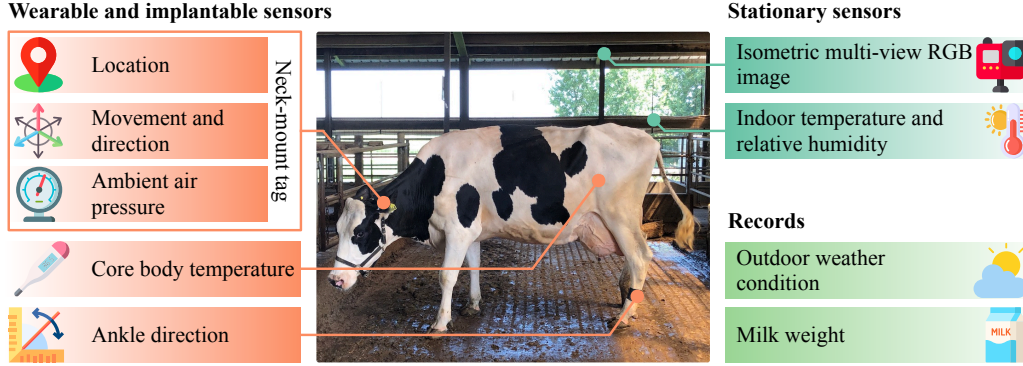

Figure 2: Overview of the MMCows dataset consisting of nine sensing modalities and records.

Table 2: Properties of nine modalities in MMCows.

| Name | Sensor | Interval | # of sensors | Primary data produced |
|------|--------|----------|--------------|----------------------|
| uwb | UWB | 15 s | 10 | Location in 3D |
| immu | IMMU | 100 ms | 10 | Head movement and direction |
| pressure | Pressure sensor | 100 ms | 10 | Air pressure |
| cbt | Vaginal temp. sensor | 1 min | 10 | CBT |
| ankle | Ankle sensor | 1 min | 10 | Ankle direction |
| rgb | Camera | 1 s | 4 | RGB image sequences |
| thi | Microclimate sensor | 1 min | 6 | Indoor temperature and RH |
| weather | Weather information | 5 min | - | Outdoor weather condition |
| milk | Milk yield record | 1 day | 1 | Total milk weight |

## 3 The MMCows Dataset

MMCows is the most comprehensive cattle monitoring dataset, consisting of the largest number of sensing modalities. To achieve this, we designed, implemented, and deployed a multimodal sensor system composed of various wearable and stationary sensors for long-term synchronized measurement. This section describes how the data in MMCows is collected, processed, and annotated.

### 3.1 Data collection and processing

We present various types of sensors used to build the dataset, selected for their effectiveness in measuring the chosen behavioral and physiological factors. Here, we discuss our custom-built sensors alongside widely used, practical, commercial-off-the-shelf sensors for measuring location, posture, movement, body temperature, etc. We also collect microclimate, weather, and milk production data, as these also play a critical role in monitoring dairy cattle health. We discuss in detail the configuration, calibration, and synchronization of each sensor.

**Sensing modalities.** In cattle monitoring, identification, localization, behavior classification, and CBT monitoring are commonly required. For this, we have chosen five wearable and implantable sensors for behavioral and physiological sensing: UWB, inertial and magnetic measurement unit (IMMU), pressure sensor, vaginal temperature sensor, and ankle sensor, as shown in Figure 2. The IMMU contains a magnetic sensor in addition to the conventional IMU. For image data collection, we use wall-mounted RGB cameras. Finally, since dairy cattle behavior and milk yield are closely related to weather conditions, we deploy indoor microclimate sensors, record outdoor weather information, and collect logs of daily milk yield. Table 2 lists these nine modalities.

**Experimental settings.** The data were collected at the Arlington Agricultural Research Station of the University of Wisconsin–Madison, Wisconsin, United States, for two weeks from July 21st to August 4th, 2023. We selected a duration with predicted heat waves to observe heat stress-related data. Figure 4 shows the top-view map of a 20×12 m pen and the installation locations of UWB anchors, cameras, and microclimate sensors, as well as a view from Camera 4. The pen houses 16

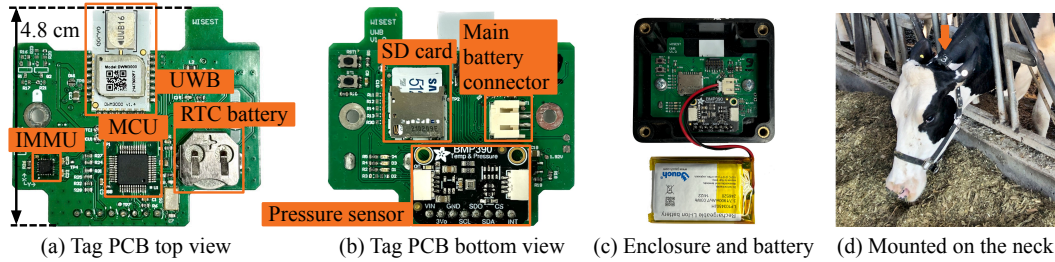

(a) Tag PCB top view    (b) Tag PCB bottom view    (c) Enclosure and battery    (d) Mounted on the neck

Figure 3: (a) Top and (b) bottom view of the wearable neck-mount tag with UWB, IMMU, and pressure sensor. (c) Water-proof enclosure with a battery. (d) The tag is mounted at the top of the neck using a neck halter.

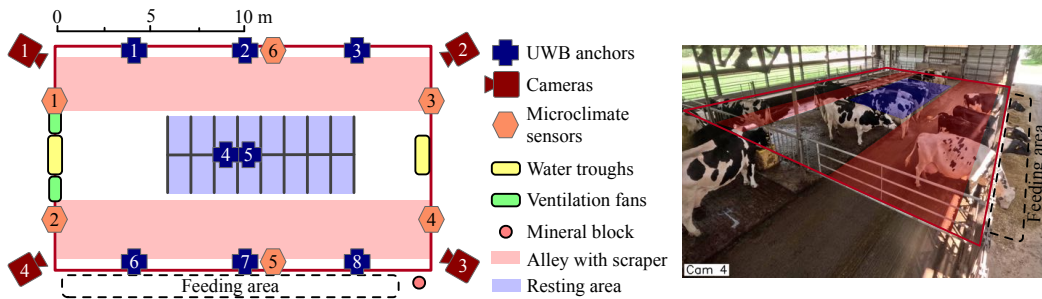

Figure 4: Top-view map of the pen with installation locations of the UWB anchors, cameras, and microclimate sensors.

Holstein cows, ten of which are equipped with wearable sensors, described next. The cattle stay in the pen most of the time except for milking twice per day for approximately 30 minutes each time. All procedures were performed with the approval of the Institutional Animal Care and Use Committee (IACUC) of the University of Wisconsin–Madison (Protocol #A006606).

**Neck-mount tag (`uwb`, `immu`, `pressure`).** We designed a wearable neck-mount tag to measure individual cows' locations and behavior. Figure 3 shows the tag's PCB and how it is mounted on the cow. The tag consists of (1) the Qorvo DW3000 UWB module for measuring distances from the tag to eight stationary UWB anchors, which are used to derive 3D neck location of the cow, (2) the TDK ICM-20984 IMMU that measures acceleration and magnetic field, and (3) the Bosch BMP390 air pressure sensor for measuring elevation. To minimize measurement noise, the UWB, the accelerometer inside the IMMU, and the pressure sensor are configured to perform oversampling at rates of 5x, 16x, and 8x, respectively. Refer to Table 2 for the sampling configurations. The tag, along with a 1.9-Ah lithium battery, is enclosed in a water-proof casing with an air-permeable hole for air pressure measurement. As shown in Figure 4, eight UWB anchors are installed at a height of 5 m to enable 3D spatial measurements. Ten of these tags are attached to ten cows (Cow 1 through Cow 10) out of a group of 16. Two additional tags are mounted in stationary positions around the pen to record data as reference points.

**Vaginal temperature sensor (`cbt`).** Each of the same ten cows is equipped with an Onset HOBO U12-15 temperature logger, inserted in their vagina to measure the vaginal temperature, which is considered the CBT of the cows.

**Ankle sensor (`ankle`).** As a common approach to detecting lying behavior, we attached an Onset HOBO Pendant G ankle sensor to the left hind leg of the same cows. This sensor measures the direction of gravity, which allows us to infer the orientation of the leg.

**Cameras (`rgb`).** Four GoPro HERO11 Black cameras are mounted at four corners of the pen, directed toward the center, as shown in Figure 4. The cameras record 4480×2800 (4.5K) videos at 1 fps, providing isometric views of the pen instead of top views, which are more common in many datasets. Detection and identification tasks become much more challenging with isometric-view images, but

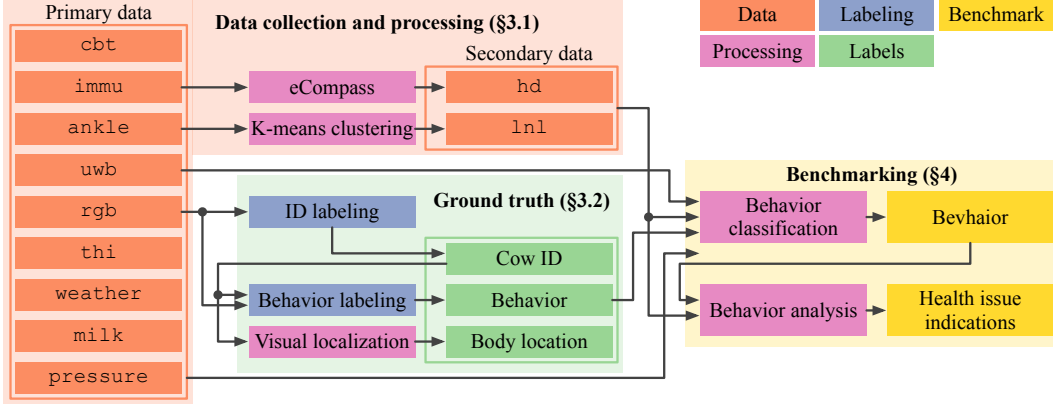

Figure 5: The data processing and benchmarking pipeline of MMCows in this paper. MMCows can be used in different processing pipelines for different applications.

this setup is more common in commercial barns as it provides wider visibility. We do not add any additional lighting to reduce motion blur, in order to maintain the same barn environment.

**Indoor microclimate (`thi`) and outdoor weather (`weather`).** The ambient temperature and relative humidity (RH) are used to infer the Temperature-Humidity Index (THI) [71], and are measured by six Onset HOBO Pro v2 temperature/RH loggers deployed as shown in Figure 4. The outdoor weather conditions are recorded every 5 minutes by a weather station located 950 m northeast of the experimental site, which includes dew point, precipitation, sunlight intensity, wind speed and direction, etc.

**Milk yield record (`milk`).** The dataset also contains the daily milk yield in kg and the health checkup records of all 16 cows prior to and after the experiment, which are recorded by the barn staff.

**Secondary data.** We generate secondary data based on the collected primary data. Secondary data in MMCows includes head direction (`hd`) from IMMU, and lying/non-lying classification (`lnl`) using the ankle sensor data. The processing pipeline for secondary data generation is illustrated in Figure 5. Note that this pipeline is only an example used in this paper, and the dataset can be used in different processing pipelines for various applications.

**Sensor calibration.** Ten UWB modules in the tags and eight UWB anchors are pairwise calibrated from 0 to 18 m using a linear calibration model. The accelerometer in the IMMU is also separately calibrated using linear offset, and the magnetometer is calibrated for hard and soft iron biases [72]. The MCU stores the calibration parameters and corrects the measurements of the IMMU in real time, while the pressure sensor is programmed to perform self-calibration upon powering on.

**Sensor synchronization.** Each neck tag is equipped with the Analog Devices DS3231 real-time clock to maintain its local time. The local time of all neck-mounted tags is synchronized with Internet time every 30 minutes through a stationary hub using UWB communication. The hub contains a Raspberry Pi 4 connected to the Internet to retrieve and retain the Internet time. In other words, the timestamps of all data collected by different sensors in the tags are synchronized with Internet time. Every 15 seconds, the tags synchronously perform distance measurements one by one in succession to prevent packet collisions when using the same wireless channel.

## 3.2 Ground truth

### 3.2.1 Ground truth for cow identification and behavior classification

The ground truth of cow IDs and behaviors is manually created using the vision data. The UWB-synced RGB frames from July 25th are used for the annotation. We manually annotated 20,000 images to produce 213,000 bounding boxes for all 16 cows. When the cows are lying in the stalls, their bodies are often heavily occluded by one another, making identification more challenging. Thus, we separate the bounding box labels of cows into three sets: lying, non-lying, and combined labels. The annotators were trained to follow our comprehensive annotation rules to ensure consistency.

The correctness of cow ID annotation was automatically verified using visual localization, which is elaborated in Section 3.2.2, and was also manually verified during the process of behavior labeling. Details of the annotation rules are provided on the dataset website.

We manually created the behavior ground truth of 16 cows at the granularity of one-second intervals, where each behavior label of the cow is associated with a timestamp. The synchronization of RGB frames from four cameras ensures that for each given timestamp, each cow performs the same behavior in all camera views. We use seven behaviors of individual cows, including walking, standing, feeding head up, feeding head down, licking, drinking, and lying, which are commonly used in various cattle behavior monitoring studies discussed in Section 2.1. Details of behavior definitions and visual examples are available in the supplementary document.

### 3.2.2 Visual localization and location ground truth

To provide reliable ground truth for cows' locations, we propose a new optimization-based approach to calculate body location using the annotated bounding boxes from multiple views. We first project the bounding box centers of the same cow across multiple views into the world coordinate system as 3D lines that inherently converge in space. We then apply AdaGrad [73] to find the optimal location that is nearest to the lines, which serves as the cow's location in 3D. This location can be used as ground truth for developing vision-based localization models. More details are provided in the supplementary document.

As an additional step to ensure the correctness of the ID annotation process, for each location, we calculate the distance from that location to its corresponding projection lines. Any line that is irregularly far from the visual location is flagged as an outlier, indicating that the cow's ID has been incorrectly annotated, which is subsequently corrected until all projection lines converge.

## 4 Evaluation and Benchmarks

For the evaluation of MMCows, we conduct a two-stage benchmarking process. In the first stage, to show how the multimodal dataset can be used for system design, we compare different modalities for the behavior classification task. In the second stage, we perform a high-level behavior analysis to show how MMCows can be used for automated dairy barn management. As this section only briefly introduces the results, we discuss them in more detail in the supplementary document.

### 4.1 Modality comparison for behavior classification

**Setting.** To avoid data leakage, we consider two data split settings: object-wise split (OS) and temporal split (TS). The OS setting evaluates cross-cow generalization, while the TS setting assesses robustness on unseen data. For `uwb` and `immu`, data from ten cows with tags is used, and both OS and TS settings are applied. For `rgb`, data from all 16 cows is used, and only the TS setting is applied since data from all cows is required to train the identification models.

In the OS setting, we use 5-fold cross-validation to train models, with the ratio of cows for training, validation, and testing set to [6:2:2]. The selection of cows is rotated so that each cow appears in the test set exactly once.

In the TS setting, data from each modality is separated into two groups based on the lighting conditions. The first group contains data with artificial light, recorded between 3am–6am and 6pm–12am. The second group contains data with natural light, recorded between 6am–6pm. Each group is divided equally and temporally into five segments, resulting in ten segments. We apply 5-fold cross-validation, where from each group, the ratio of segments for training, validation, and testing is set to [3:1:1]. The two groups are concatenated in the configuration [6:2:2]. Validation is performed until each segment has been tested once.

**Metric.** Cows often spend substantial amounts of time lying, feeding, and standing, but only seldom engaging in walking, drinking, or licking. Since the behavior classes are heavily imbalanced and the minority classes are also important, we use the F1 score to evaluate the results. An average F1 score is reported for each setting.

**Methods.** We perform behavior classification using `uwb`, `immu`, `rgb`, and some combinations of them with `hd` and `ankle`. We selected three specific combinations of modalities to demonstrate the benefits

Table 3: Performance comparison of behavior classification using different modalities.

| Modality | Setting | F1 score ↑ | | | | | | | |
|---|---|---|---|---|---|---|---|---|---|
| | | Walking | Standing | Feeding↑ | Feeding↓ | Licking | Drinking | Lying | Average |
| UWB | OS | .078±.027 | .855±.023 | .704±.077 | .834±.049 | **.884**±.054 | .644±.112 | .953±.017 | .707±.051 |
| | TS | .103±.040 | **.860**±.041 | **.738**±.026 | **.835**±.029 | .868±.066 | **.656**±.059 | **.961**±.008 | **.717**±.038 |
| IMMU | OS | .000±.000 | .065±.127 | .067±.084 | .098±.135 | .000±.000 | .000±.000 | .700±.760 | .133±.060 |
| | TS | .000±.000 | .052±.053 | .000±.000 | .051±.048 | .000±.000 | .000±.000 | .742±.126 | .141±.038 |
| RGBs | TS | **.143**±.036 | .814±.048 | .634±.063 | .715±.051 | .484±.193 | .409±.116 | .681±.032 | .554±.077 |
| UWB+HD | OS | .032±.030 | .908±.015 | .731±.059 | .843±.046 | .812±.154 | .645±.136 | .980±.006 | .707±.064 |
| | TS | .074±.036 | .917±.022 | .766±.030 | .853±.026 | **.863**±.057 | **.699**±.049 | .986±.003 | .737±.032 |
| UWB+HD+Akl | OS | .048±.040 | .937±.014 | .730±.057 | .842±.044 | .800±.183 | .643±.132 | .996±.001 | .714±.067 |
| | TS | .055±.026 | **.938**±.014 | **.768**±.032 | **.854**±.023 | **.863**±.060 | .684±.041 | **.997**±.001 | **.737**±.028 |
| RGBm | TS | **.127**±.053 | .815±.030 | .741±.044 | .805±.046 | .578±.172 | .478±.154 | .883±.027 | .632±.075 |

of incorporating multiple modalities to enhance the robustness of behavior monitoring. While UWB alone is sufficient for detecting most behaviors, it lacks the precision needed to distinguish some similar behaviors, such as whether a cow is merely standing or engaging in other activities at the same location. Adding head direction helps differentiate between behaviors like standing versus drinking, standing versus licking, and feeding head up versus feeding head down. Additionally, since cows often stand in the stalls where their head direction can change over time, integrating ankle acceleration provides a clear distinction between standing and lying.

- UWB: Random Forest (RF) with balanced weights. The `uwb` data is used.

- IMMU: Fully connected network. The accelerometer and magnetic sensor data in `immu` are used. The discrete wavelet transform is applied to the accelerometer data. The data is segmented using 10-second window with 50% overlap, and the midpoint behavior is selected as the label of each window.

- RGBs: Three-stage pipeline that consists of YOLOv8 [74] for cow detection and EfficientNet-B0 [75] for behavior classification and cow identification using single-view images from the `rgb` data. Two separate models are trained for identifying lying and non-lying cows.

- UWB+HD: RF with balanced weights. The combination of `uwb` and `hd` data is used.

- UWB+HD+Akl: RF with balanced weights. The combination of `uwb`, `hd`, and `ankle` data is used.

- RGBm: Multi-view processing of the `rgb` data. At each time point, four outputs of the RGBs method are projected into the world coordinate system as 3D lines, where outliers are detected and excluded. The remaining projection lines are combined using weighted voting, where each weight is proportional to the width of its corresponding bounding box.

**Results and discussion.** Table 3 shows the F1 scores of behavior classification. Overall, all models achieve the highest F1 scores for lying, followed by standing, licking, feeding head down, and feeding head up. The F1 scores are slightly higher under TS compared to OS for all models. Walking is hard to distinguish from standing as we use image frames individually, but this would be greatly improved if the data were processed as sequences of frames. UWB and RGBs outperform IMMU for most behaviors except walking. Combining multiple modalities substantially reduces errors. The best model is UWB+HD+Akl, illustrating the complementary benefits of multimodal ML. RGBm exhibits a noticeable improvement over RGBs in all behavior classes as knowledge from multiple views is combined. The results clearly show the varying performance of different modalities for the same task, which proves the value of our multimodal dataset.

## 4.2 Behavior analysis

We show how MмCows can be used to develop ML models for dairy cattle health monitoring.

**Setting.** As standing, lying, feeding, and drinking behaviors are strongly correlated with THI, we analyze changes in these behaviors and correlate them with the `thi` data. We evaluate the number of bouts, mean duration of bouts, and total duration of each behavior per day, obtained from behavior

Table 4: Correlations between daily [frequency of bouts, mean duration of bouts, total duration] of cows' behaviors versus the THI for 13 days.

| Behavior | $(|r| > 0)$ ↑ | $p$-value ↓ | $R^2$ ↑ |
|---|---|---|---|
| Standing | [–.470, **.726**, **.678**] | [.105, **.005**, **.011**] | [.221, **.527**, **.459**] |
| Lying | [ .476, **–.744**, **–.696**] | [.158, **.004**, **.008**] | [.173, **.553**, **.484**] |
| Feeding | [ **.502**, **–.581**, –.424] | [.080, **.037**, .149] | [.252, .338, .180] |
| Drinking | [ **.802**, .196, .476] | [**.001**, .521, .100] | [**.643**, .038, .227] |

classification over 13 days with respect to THI (excluding the first and last days that are shorter than 24 hours).

**Metric.** We use the Pearson correlation coefficient ($r$), $p$-value, and $R^2$ to evaluate the relationship between variables. The $r$ value helps to identify the strength and direction of the relationship; the $p$-value determines the statistical significance, and $R^2$ indicates the proportion of variability explained by the THI.

**Method.** We use UWB+HD+Akl to perform inference on 2-week-long data to extract seven behaviors of ten cows. Feeding head up and head down behaviors are combined as feeding. To obtain an accurate number of bouts for each behavior, we use a custom filter to remove momentary switching between classes.

**Results and discussion.** We confirm significant correlations between the cows' behavior and THI as reported in Table 4. All behaviors are strongly affected by THI, indicated by the very small $p$-values and high $R^2$ values. The results agree with previous studies on THI-dependent changes in cattle behavior [16, 17, 18, 76, 77, 78, 79]. The results show the effectiveness of behavior monitoring in assessing cattle health status, such as heat stress.

# 5  Conclusion

MMCows is the first multimodal dataset for dairy cattle monitoring, comprising nine modalities and records. It integrates wearable, implantable, visual, and environmental data collected from 16 milking Holstein cows in a real-world barn over two weeks. This paper describes the creation of this dataset and demonstrates its potential benefits for developing monitoring devices and ML models. The true potential of MMCows lies in the numerous combinations of modalities that have yet to be explored, which can contribute to designing high-accuracy, low-cost, and animal-friendly monitoring systems. We envision MMCows being leveraged for a variety of future endeavors, to promote environmentally, socially, and economically sustainable dairy farming.

# Acknowledgements

The authors thank the anonymous reviewers for their valuable feedback. This work was supported by the USDA National Institute of Food and Agriculture grant 2021-67021-34036 and the National Science Foundation grant 2435327.

# References

[1] D. O'Brien, J. L. Capper, P. C. Garnsworthy, C. Grainger, and L. Shalloo. A case study of the carbon footprint of milk from high-performing confinement and grass-based dairy farms. *Journal of Dairy Science*, 97(3):1835–1851, 2014.

[2] Clarissa S. Cardoso, Maria José Hötzel, Daniel M. Weary, Jesse A. Robbins, and Marina A.G. von Keyserlingk. Imagining the ideal dairy farm. *Journal of Dairy Science*, 99(2):1663–1671, 2016.

[3] H. W. Barkema, M. A. G. von Keyserlingk, J. P. Kastelic, T. J. G. M. Lam, C. Luby, J. P. Roy, S. J. LeBlanc, G. P. Keefe, and D. F. Kelton. Invited review: Changes in the dairy industry affecting dairy cattle health and welfare. *Journal of Dairy Science*, 98(11):7426–7445, 2015.

[4] Daniela Lovarelli, Jacopo Bacenetti, and Marcella Guarino. A review on dairy cattle farming: Is precision livestock farming the compromise for an environmental, economic and social sustainable production? *Journal of Cleaner Production*, 262:121409, 2020.

[5] Emanuela Tullo, Alberto Finzi, and Marcella Guarino. Review: Environmental impact of livestock farming and precision livestock farming as a mitigation strategy. *Science of The Total Environment*, 650:2751–2760, 2019.

[6] Philip Thornton, Gerald Nelson, Dianne Mayberry, and Mario Herrero. Impacts of heat stress on global cattle production during the 21st century: A modelling study. *The Lancet Planetary Health*, 6(3):e192–e201, 2022.

[7] Suresh Neethirajan and Bas Kemp. Digital livestock farming. *Sensing and Bio-Sensing Research*, 32:100408, 2021.

[8] Jiangjing Liu, Lanqi Li, Xiaoli Chen, Yongqiang Lu, and Dong Wang. Effects of heat stress on body temperature, milk production, and reproduction in dairy cows: A novel idea for monitoring and evaluation of heat stress—A review. *Asian-Australasian Journal of Animal Sciences*, 32(9):1332, 2019.

[9] Sébastien Fournel, Alain N. Rousseau, and Benoit Laberge. Rethinking environment control strategy of confined animal housing systems through precision livestock farming. *Biosystems Engineering*, 155:96–123, 2017.

[10] Rodrigo García, Jose Aguilar, Mauricio Toro, Angel Pinto, and Paul Rodríguez. A systematic literature review on the use of machine learning in precision livestock farming. *Computers and Electronics in Agriculture*, 179:105826, 2020.

[11] Md Sultan Mahmud, Azlan Zahid, Anup Kumar Das, Muhammad Muzammil, and Muhammad Usman Khan. A systematic literature review on deep learning applications for precision cattle farming. *Computers and Electronics in Agriculture*, 187:106313, 2021.

[12] A. Cominotte, A. F. A. Fernandes, J. R. R. Dorea, G. J. M. Rosa, M. M. Ladeira, E. H. C. B. van Cleef, G. L. Pereira, W. A. Baldassini, and O. R. Machado Neto. Automated computer vision system to predict body weight and average daily gain in beef cattle during growing and finishing phases. *Livestock Science*, 232:103904, 2020.

[13] Yigit Tuncel, Toygun Basaklar, Mackenzie Smithyman, João Dórea, Vinícius Nunes De Gouvêa, Younghyun Kim, and Umit Ogras. Advancing cattle welfare: Ultra low-power health monitoring at the edge. In *IEEE Biomedical Circuits and Systems Conference (BioCAS)*, pages 1–5, 2023.

[14] William Andrew, Jing Gao, Siobhan Mullan, Neill Campbell, Andrew W Dowsey, and Tilo Burghardt. Visual identification of individual holstein-friesian cattle via deep metric learning. *Computers and Electronics in Agriculture*, 185:106133, 2021.

[15] C. A. Becker, R. J. Collier, and A. E. Stone. Invited review: Physiological and behavioral effects of heat stress in dairy cows. *Journal of Dairy Science*, 103(8):6751–6770, 2020.

[16] N. B. Cook, R. L. Mentink, T. B. Bennett, and K. Burgi. The effect of heat stress and lameness on time budgets of lactating dairy cows. *Journal of Dairy Science*, 90(4):1674–1682, 2007.

[17] J. D. Allen, L. W. Hall, R. J. Collier, and J. F. Smith. Effect of core body temperature, time of day, and climate conditions on behavioral patterns of lactating dairy cows experiencing mild to moderate heat stress. *Journal of Dairy Science*, 98(1):118–127, 2015.

[18] Grazyne Tresoldi, Karin E. Schütz, and Cassandra B. Tucker. Cooling cows with sprinklers: Effects of soaker flow rate and timing on behavioral and physiological responses to heat load and production. *Journal of Dairy Science*, 102(1):528–538, 2019.

[19] Abdul Sammad, Ya Jing Wang, Saqib Umer, Hu Lirong, Imran Khan, Adnan Khan, Baseer Ahmad, and Yachun Wang. Nutritional physiology and biochemistry of dairy cattle under the influence of heat stress: Consequences and opportunities. *Animals*, 10(5):793, 2020.

[20] Charles T. Kadzere, Michael R. Murphy, Nissim Silanikove, and Elliot Maltz. Heat stress in lactating dairy cows: A review. *Livestock Production Science*, 77(1):59–91, 2002.

[21] R. J. Collier, B. J. Renquist, and Y. Xiao. A 100-year review: Stress physiology including heat stress. *Journal of Dairy Science*, 100(12):10367–10380, 2017.

[22] P. R. Hut, J. Scheurwater, M. Nielen, J. Van Den Broek, and M. M. Hostens. Heat stress in a temperate climate leads to adapted sensor-based behavioral patterns of dairy cows. *Journal of Dairy Science*, 105(8):6909–6922, 2022.

[23] Jessica B. Wheelock, Robert P. Rhoads, Matthew J. VanBaale, Susan R. Sanders, and Lance H. Baumgard. Effects of heat stress on energetic metabolism in lactating holstein cows. *Journal of Dairy Science*, 93(2):644–655, 2010.

[24] G. André, B. Engel, P. B. M. Berentsen, Th. V. Vellinga, and A. G. J. M. Oude Lansink. Quantifying the effect of heat stress on daily milk yield and monitoring dynamic changes using an adaptive dynamic model. *Journal of Dairy Science*, 94(9):4502–4513, 2011.

[25] A. E. Stone, B. W. Jones, C. A. Becker, and J. M. Bewley. Influence of breed, milk yield, and temperature-humidity index on dairy cow lying time, neck activity, reticulorumen temperature, and rumination behavior. *Journal of Dairy Science*, 100(3):2395–2403, 2017.

[26] Elena Galán, Pol Llonch, Arantxa Villagrá, Harel Levit, Severino Pinto, and Agustín Del Prado. A systematic review of non-productivity-related animal-based indicators of heat stress resilience in dairy cattle. *PloS One*, 13(11):e0206520, 2018.

[27] Nigel B. Cook and Kenneth V. Nordlund. The influence of the environment on dairy cow behavior, claw health and herd lameness dynamics. *The Veterinary Journal*, 179(3):360–369, 2009.

[28] L. A. González, B. J. Tolkamp, M. P. Coffey, A. Ferret, and I. Kyriazakis. Changes in feeding behavior as possible indicators for the automatic monitoring of health disorders in dairy cows. *Journal of Dairy Science*, 91(3):1017–1028, 2008.

[29] Vivi M. Thorup, Birte L. Nielsen, Pierre-Emmanuel Robert, Sylvie Giger-Reverdin, Jakub Konka, Craig Michie, and Nicolas C. Friggens. Lameness affects cow feeding but not rumination behavior as characterized from sensor data. *Frontiers in Veterinary Science*, 3:37, 2016.

[30] G. M. Borghart, L. E. O'Grady, and J. R. Somers. Prediction of lameness using automatically recorded activity, behavior and production data in post-parturient irish dairy cows. *Irish Veterinary Journal*, 74:1–10, 2021.

[31] Nathalie Bareille, F. Beaudeau, Stéphanie Billon, A. Robert, and Philippe Faverdin. Effects of health disorders on feed intake and milk production in dairy cows. *Livestock Production Science*, 83(1):53–62, 2003.

[32] Katrine Kop Fogsgaard, Christine Maria Røntved, Peter Sørensen, and Mette S Herskin. Sickness behavior in dairy cows during escherichia coli mastitis. *Journal of Dairy Science*, 95(2):630–638, 2012.

[33] P. Løvendahl and M. G. G. Chagunda. On the use of physical activity monitoring for estrus detection in dairy cows. *Journal of Dairy Science*, 93(1):249–259, 2010.

[34] Said Benaissa, Frank André Maurice Tuyttens, David Plets, Jens Trogh, Luc Martens, Leen Vandaele, Wout Joseph, and Bart Sonck. Calving and estrus detection in dairy cattle using a combination of indoor localization and accelerometer sensors. *Computers and Electronics in Agriculture*, 168:105153, 2020.

[35] Ephraim Maltz and Aharon Antler. A practical way to detect approaching calving of the dairy cow by a behaviour sensor. In *Precision Livestock Farming*, pages 141–146. 2007.

[36] K. Schirmann, N. Chapinal, D. M. Weary, L. Vickers, and M. A. G. Von Keyserlingk. Rumination and feeding behavior before and after calving in dairy cows. *Journal of Dairy Science*, 96(11):7088–7092, 2013.

[37] L. N. Grinter, M. R. Campler, and J. H. C. Costa. Validation of a behavior-monitoring collar's precision and accuracy to measure rumination, feeding, and resting time of lactating dairy cows. *Journal of Dairy Science*, 102(4):3487–3494, 2019.

[38] Weizheng Shen, Fei Cheng, Yu Zhang, Xiaoli Wei, Qiang Fu, and Yonggen Zhang. Automatic recognition of ingestive-related behaviors of dairy cows based on triaxial acceleration. *Information Processing in Agriculture*, 7(3):427–443, 2020.

[39] Jose M. Chapa, Kristina Maschat, Michael Iwersen, Johannes Baumgartner, and Marc Drillich. Accelerometer systems as tools for health and welfare assessment in cattle and pigs – A review. *Behavioural Processes*, 181:104262, 2020.

[40] Kim Margarette Corpuz Nogoy, Sun-il Chon, Ji-hwan Park, Saraswathi Sivamani, Dong-Hoon Lee, and Seong Ho Choi. High precision classification of resting and eating behaviors of cattle by using a collar-fitted triaxial accelerometer sensor. *Sensors*, 22(16):5961, 2022.

[41] B. Wolfger, B. W. Jones, K. Orsel, and J. M. Bewley. Evaluation of an ear-attached real-time location monitoring system. *Journal of Dairy Science*, 100(3):2219–2224, 2017.

[42] Bruno Meunier, Philippe Pradel, Karen H. Sloth, Carole Cirié, Eric Delval, Marie M. Mialon, and Isabelle Veissier. Image analysis to refine measurements of dairy cow behaviour from a real-time location system. *Biosystems Engineering*, 173:32–44, 2018.

[43] E. M. Homer, Y. Gao, X. Meng, A. Dodson, R. Webb, and P. C. Garnsworthy. A novel approach to the detection of estrus in dairy cows using ultra-wideband technology. *Journal of Dairy Science*, 96(10):6529–6534, 2013.

[44] Keni Ren, Gun Bernes, Mårten Hetta, and Johannes Karlsson. Tracking and analysing social interactions in dairy cattle with real-time locating system and machine learning. *Journal of Systems Architecture*, 116:102139, 2021.

[45] Hai Wang, Abraham O. Fapojuwo, and Robert J. Davies. A wireless sensor network for feedlot animal health monitoring. *IEEE Sensors Journal*, 16(16):6433–6446, 2016.

[46] Brahim Achour, Malika Belkadi, Idir Filali, Mourad Laghrouche, and Mourad Lahdir. Image analysis for individual identification and feeding behaviour monitoring of dairy cows based on Convolutional Neural Networks (CNN). *Biosystems Engineering*, 198:31–49, 2020.

[47] Hang Shu, Jérôme Bindelle, Leifeng Guo, and Xianhong Gu. Determining the onset of heat stress in a dairy herd based on automated behaviour recognition. *Biosystems Engineering*, 226:238–251, 2023.

[48] William Andrew, Colin Greatwood, and Tilo Burghardt. Visual localisation and individual identification of holstein friesian cattle via deep learning. In *Proceedings of the IEEE International Conference on Computer Vision Workshops (ICCV)*, pages 2850–2859, 2017.

[49] Ramendra Das, Lalrengpuii Sailo, Nishant Verma, Pranay Bharti, Jnyanashree Saikia, and Rakesh Kumar. Impact of heat stress on health and performance of dairy animals: A review. *Veterinary World*, 9(3):260, 2016.

[50] Ian K. Atkins, Nigel B. Cook, Mario R. Mondaca, and Christopher Y. Choi. Continuous respiration rate measurement of heat-stressed dairy cows and relation to environment, body temperature, and lying time. *Transactions of the American Society of Agricultural and Biological Engineers (ASABE)*, 61(5):1475–1485, 2018.

[51] Musadiq Idris, Jashim Uddin, Megan Sullivan, David M. McNeill, and Clive J. C. Phillips. Non-invasive physiological indicators of heat stress in cattle. *Animals*, 11(1):71, 2021.

[52] Soraia F. Neves, Mónica C. F. Silva, João M. Miranda, George Stilwell, and Paulo P. Cortez. Predictive models of dairy cow thermal state: A review from a technological perspective. *Veterinary Sciences*, 9(8):416, 2022.

[53] smaXtec. smaxtec: Early detection for dairy cows with bolus technology. Last accessed: October 30th, 2024. URL: https://smaxtec.com/us/.

[54] Hien Vu, Hanwook Chung, Christopher Choi, and Younghyun Kim. eTag: An energy-neutral ear tag for real-time body temperature monitoring of dairy cattle. In *Proceedings of the International Conference on Mobile Computing and Networking (MobiCom)*, pages 1–15, 2023.

[55] Hanwook Chung, Hien Vu, Younghyun Kim, and Christopher Y. Choi. Subcutaneous temperature monitoring through ear tag for heat stress detection in dairy cows. *Biosystems Engineering*, 235:202–214, 2023.

[56] Hanwook Chung, Jingjie Li, Younghyun Kim, Jennifer M. C. Van Os, Sabrina H. Brounts, and Christopher Y. Choi. Using implantable biosensors and wearable scanners to monitor dairy cattle's core body temperature in real-time. *Computers and Electronics in Agriculture*, 174:105453, 2020.

[57] Hanwook Chung, Jingjie Li, Younghyun Kim, and Christopher Y. Choi. Continuous and wireless skin contact and ear implant temperature measurements and relations to the core body temperature of heat stressed dairy cows. In *Proceedings of the International Livestock Environment Symposium (ILES X)*, page 1, 2018.

[58] Katsuyuki Nakamura, Serena Yeung, Alexandre Alahi, and Li Fei-Fei. Jointly learning energy expenditures and activities using egocentric multimodal signals. In *Proceedings of the IEEE Conference on Computer Vision and Pattern Recognition (CVPR)*, pages 1868–1877, 2017.

[59] Joseph DelPreto, Chao Liu, Yiyue Luo, Michael Foshey, Yunzhu Li, Antonio Torralba, Wojciech Matusik, and Daniela Rus. Actionsense: A multimodal dataset and recording framework for human activities using wearable sensors in a kitchen environment. In *Advances in Neural Information Processing Systems (NeurIPS)*, volume 35, pages 13800–13813, 2022.

[60] Sizhe An, Yin Li, and Umit Ogras. mri: Multi-modal 3d human pose estimation dataset using mmwave, rgb-d, and inertial sensors. In *Advances in Neural Information Processing Systems (NeurIPS)*, volume 35, pages 27414–27426, 2022.

[61] Xavier Alameda-Pineda, Jacopo Staiano, Ramanathan Subramanian, Ligia Batrinca, Elisa Ricci, Bruno Lepri, Oswald Lanz, and Nicu Sebe. Salsa: A novel dataset for multimodal group behavior analysis. *IEEE Transactions on Pattern Analysis and Machine Intelligence*, 38(8):1707–1720, 2015.

[62] Dan Li, Kaifeng Zhang, Zhenbo Li, and Yifei Chen. A spatiotemporal convolutional network for multi-behavior recognition of pigs. *Sensors*, 20(8):2381, 2020.

[63] Kaifeng Zhang, Dan Li, Jiayun Huang, and Yifei Chen. Automated video behavior recognition of pigs using two-stream convolutional networks. *Sensors*, 20(4):1085, 2020.

[64] Luca Bergamini, Stefano Pini, Alessandro Simoni, Roberto Vezzani, Simone Calderara, Rick B. D. Eath, and Robert B. Fisher. Extracting accurate long-term behavior changes from a large pig dataset. In *International Joint Conference on Computer Vision, Imaging and Computer Graphics Theory and Applications (VISIGRAPP)*, pages 524–533, 2021.

[65] William Andrew, Sion Hannuna, Neill Campbell, and Tilo Burghardt. Automatic individual holstein friesian cattle identification via selective local coat pattern matching in rgb-d imagery. In *IEEE International Conference on Image Processing (ICIP)*, pages 484–488, 2016.

[66] Aram Ter-Sarkisov, Robert Ross, and John Kelleher. Bootstrapping labelled dataset construction for cow tracking and behavior analysis. In *Conference on Computer and Robot Vision (CRV)*, pages 277–284, 2017.

[67] Domingo S. Rodriguez-Baena, Francisco A. Gomez-Vela, Miguel García-Torres, Federico Divina, Carlos D. Barranco, Norberto Daz-Diaz, Manuel Jimenez, and Gema Montalvo. Identifying livestock behavior patterns based on accelerometer dataset. *Journal of Computational Science*, 41:101076, 2020.

[68] Jing Gao, Tilo Burghardt, William Andrew, Andrew W. Dowsey, and Neill W. Campbell. Towards self-supervision for video identification of individual holstein-friesian cattle: The cows2021 dataset. *arXiv Preprint arXiv:2105.01938*, 2021.

[69] Olli Koskela, Leonardo Santiago Benitez Pereira, Ilpo Pölönen, Ilmo Aronen, and Iivari Kunttu. Deep learning image recognition of cow behavior and an open data set acquired near an automatic milking robot. *Agricultural and Food Science*, 31(2):89–103, 2022.

[70] Shahid Ismail, Moises Diaz, Cristina Carmona-Duarte, Jose Manuel Vilar, and Miguel A. Ferrer. Cowscreeningdb: A public benchmark database for lameness detection in dairy cows. *Computers and Electronics in Agriculture*, 216:108500, 2024.

[71] S. Dikmen and P. J. Hansen. Is the temperature-humidity index the best indicator of heat stress in lactating dairy cows in a subtropical environment? *Journal of Dairy Science*, 92(1):109–116, 2009.

[72] MathWorks. Magnetometer calibration. Last accessed: October 30th, 2024. URL: `https://www.mathworks.com/help/nav/ug/magnetometer-calibration.html`.

[73] John Duchi, Elad Hazan, and Yoram Singer. Adaptive subgradient methods for online learning and stochastic optimization. *Journal of Machine Learning Research*, 12(7), 2011.

[74] Glenn Jocher, Ayush Chaurasia, and Jing Qiu. Ultralytics yolov8, 2023. Last accessed: October 30th, 2024. URL: `https://github.com/ultralytics/ultralytics`.

[75] Mingxing Tan and Quoc Le. Efficientnet: Rethinking model scaling for convolutional neural networks. In *International Conference on Machine Learning*, pages 6105–6114, 2019.

[76] A. Gomez and N. B. Cook. Time budgets of lactating dairy cattle in commercial freestall herds. *Journal of Dairy Science*, 93(12):5772–5781, 2010.

[77] Lisette M. C. Leliveld, Elisabetta Riva, Gabriele Mattachini, Alberto Finzi, Daniela Lovarelli, and Giorgio Provolo. Dairy cow behavior is affected by period, time of day and housing. *Animals*, 12(4):512, 2022.

[78] J. Chang-Fung-Martel, M. T. Harrison, J. N. Brown, Richard Rawnsley, A. P. Smith, and Holger Meinke. Negative relationship between dry matter intake and the temperature-humidity index with increasing heat stress in cattle: A global meta-analysis. *International Journal of Biometeorology*, 65(12):2099–2109, 2021.

[79] Yu-Chi Tsai, Jih-Tay Hsu, Shih-Torng Ding, Dan Jeric Arcega Rustia, and Ta-Te Lin. Assessment of dairy cow heat stress by monitoring drinking behaviour using an embedded imaging system. *Biosystems Engineering*, 199:97–108, 2020.

